# Variational Inference over Combinatorial Spaces

**Alexandre Bouchard-Côté**[*]    **Michael I. Jordan**[*,†]
[*]Computer Science Division    [†]Department of Statistics
University of California at Berkeley

## Abstract

Since the discovery of sophisticated fully polynomial randomized algorithms for a range of #P problems [1, 2, 3], theoretical work on approximate inference in combinatorial spaces has focused on Markov chain Monte Carlo methods. Despite their strong theoretical guarantees, the slow running time of many of these randomized algorithms and the restrictive assumptions on the potentials have hindered the applicability of these algorithms to machine learning. Because of this, in applications to combinatorial spaces simple exact models are often preferred to more complex models that require approximate inference [4]. Variational inference would appear to provide an appealing alternative, given the success of variational methods for graphical models [5]; unfortunately, however, it is not obvious how to develop variational approximations for combinatorial objects such as matchings, partial orders, plane partitions and sequence alignments. We propose a new framework that extends variational inference to a wide range of combinatorial spaces. Our method is based on a simple assumption: the existence of a tractable *measure factorization*, which we show holds in many examples. Simulations on a range of matching models show that the algorithm is more general and empirically faster than a popular fully polynomial randomized algorithm. We also apply the framework to the problem of multiple alignment of protein sequences, obtaining state-of-the-art results on the BAliBASE dataset [6].

## 1   Introduction

The framework we propose is applicable in the following setup: let $\mathcal{C}$ denote a *combinatorial space*, by which we mean a finite but large set, where testing membership is tractable, but enumeration is not, and suppose that the goal is to compute $\sum_{x \in \mathcal{C}} f(x)$, where $f$ is a positive function. This setup subsumes many probabilistic inference and classical combinatorics problems. It is often intractable to compute this sum, so approximations are used.

We approach this problem by exploiting a finite collection of sets $\{\mathcal{C}_i\}$ such that $\mathcal{C} = \cap_i \mathcal{C}_i$. Each $\mathcal{C}_i$ is larger than $\mathcal{C}$, but paradoxically it is often possible to find such a decomposition where for each $i$, $\sum_{x \in \mathcal{C}_i} f(x)$ is tractable. We give many examples of this in Section 3 and Appendix B.[1] This paper describes an effective way of using this type of decomposition to approximate the original sum.

Another way of viewing this setup is in terms of exponential families. In this view, described in detail in Section 2, the decomposition becomes a factorization of the base measure. As we will show, the exponential family view gives a principled way of defining variational approximations. In order to make variational approximations tractable in the combinatorial setup, we use what we call an *implicit message representation*. The canonical parameter space of the exponential family enables such representation. We also show how additional approximations can be introduced in cases where the factorization has a large number of factors. These further approximations rely on an outer bound of the partition function, and therefore preserve the guarantees of convex variational objective functions.

While previous authors have proposed mean field or loopy belief propagation algorithms to approximate the partition function of a few specific combinatorial models—for example [7, 8] for parsing,

and [9, 10] for computing the permanent of a matrix—we are not aware of a general treatment of variational inference in combinatorial spaces.

There has been work on applying variational algorithms to the problem of *maximization* over combinatorial spaces [11, 12, 13, 14], but maximization over combinatorial spaces is rather different than summation. For example, in the bipartite matching example considered in both [13] and this paper, there is a known polynomial algorithm for maximization, but not for summation. Our approach is also related to agreement-based learning [15, 16], although agreement-based learning is defined within the context of unsupervised learning using EM, while our framework is agnostic with respect to parameter estimation.

The paper is organized as follows: in Section 2 we present the measure factorization framework; in Section 3 we show examples of this framework applied to various combinatorial inference problems; and in Section 4 we present empirical results.

## 2 Variational measure factorization

In this section, we present the variational measure factorization framework. At a high level, the first step is to construct an equivalent but more convenient exponential family. This exponential family will allow us to transform variational algorithms over graphical models into approximation algorithms over combinatorial spaces. We first describe the techniques needed to do this transformation in the case of a specific variational inference algorithm—loopy belief propagation—and then discuss mean-field and tree-reweighted approximations.

To make the exposition more concrete, we use the running example of approximating the value and gradient of the log-partition function of a Bipartite Matching model (BM) over $K_{N,N}$, a well-known #P problem [17]. Unless we mention otherwise, we will consider bipartite perfect matchings; non-bipartite and non-perfect matchings are discussed in Section 3.1. The reader should keep in mind, however, that our framework is applicable to a much broader class of combinatorial objects. We develop several other examples in Section 3 and in Appendix B.

### 2.1 Setup

Since we are dealing with discrete-valued random variables $\boldsymbol{X}$, we can assume without loss of generality that the probability distribution for which we want to compute the partition function and moments is a member of a regular exponential family with canonical parameters $\boldsymbol{\theta} \in \mathbb{R}^J$:

$$\mathbb{P}(\boldsymbol{X} \in B) = \sum_{x \in B} \exp\{\langle \boldsymbol{\phi}(x), \boldsymbol{\theta} \rangle - A(\boldsymbol{\theta})\}\nu(x), \quad A(\boldsymbol{\theta}) = \log \sum_{x \in \boldsymbol{\mathcal{X}}} \exp\{\langle \boldsymbol{\phi}(x), \boldsymbol{\theta} \rangle\}\nu(x), \quad (1)$$

for a $J$-dimensional sufficient statistic $\phi$ and base measure $\nu$ over $\mathcal{F} = 2^{\boldsymbol{\mathcal{X}}}$, both of which are assumed (again, without loss of generality) to be indicator functions : $\phi_j, \nu : \boldsymbol{\mathcal{X}} \to \{0, 1\}$. Here $\boldsymbol{\mathcal{X}}$ is a superset of both $\mathcal{C}$ and all of the $\mathcal{C}_i$s. The link between this setup and the general problem of computing $\sum_{x \in \mathcal{C}} f(x)$ is the base measure $\nu$, which is set to the indicator function over $\mathcal{C}$: $\nu(x) = \mathbf{1}[x \in \mathcal{C}]$, where $\mathbf{1}[\cdot]$ is equal to one if its argument holds true, and zero otherwise.

The goal is to approximate $A(\boldsymbol{\theta})$ and $\nabla A(\boldsymbol{\theta})$ (recall that the $j$-th coordinate of the gradient, $\nabla_j A$, is equal to the expectation of the sufficient statistic $\phi_j$ under the exponential family with base measure $\nu$ [5]). We want to exploit situations where the base measure can be written as a product of $I$ measures $\nu(x) = \prod_{i=1}^{I} \nu_i(x)$ such that each factor $\nu_i : \boldsymbol{\mathcal{X}} \to \{0, 1\}$ induces a *super-partition function* assumed to be tractable: $A_i(\boldsymbol{\theta}) = \log \sum_{x \in \boldsymbol{\mathcal{X}}} \exp\{\langle \boldsymbol{\phi}(x), \boldsymbol{\theta} \rangle\}\nu_i(x)$. This computation is typically done using dynamic programming (DP). We also assume that the gradient of the super-partition functions is tractable, which is typical for DP formulations.

In the case of BM, the space $\boldsymbol{\mathcal{X}}$ is a product of $N^2$ binary alignment variables, $x = x_{1,1}, x_{1,2}, \ldots, x_{N,N}$. In the Standard Bipartite Matching formulation (which we denote by SBM), the sufficient statistic takes the form $\phi_j(x) = x_{m,n}$. The measure factorization we use to enforce the matching property is $\nu = \nu_1 \nu_2$, where:

$$\nu_1(x) = \prod_{m=1}^{N} \mathbf{1}[\sum_{n=1}^{N} x_{m,n} \le 1], \quad \nu_2(x) = \prod_{n=1}^{N} \mathbf{1}[\sum_{m=1}^{N} x_{m,n} \le 1]. \quad (2)$$

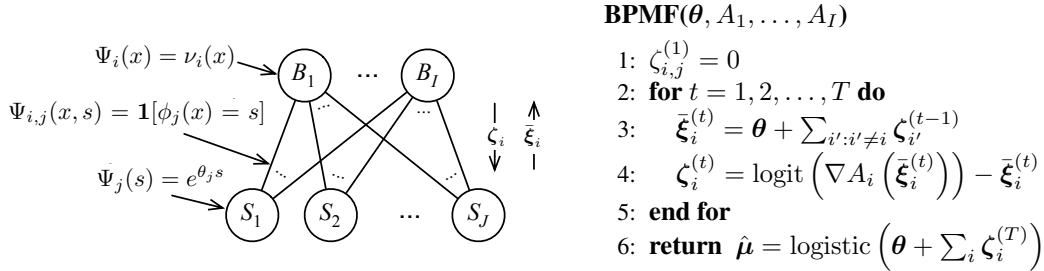

Figure 1: Left: the bipartite graphical model used for the MRF construction described in Section 2.2. Right: pseudocode for the BPMF algorithm. See Section 2 and Appendix A.2 for the derivation.

We show in Appendix A.3 that $A_1$ and $A_2$ can be computed in time $O(N^2)$ for the SBM.

The last assumption we make is that given a vector $s \in \mathbb{R}^J$, there is at most one possible configuration $x$ with $\phi(x) = s$. We call this the *rich sufficient statistic condition*. Since we are concerned in this framework with computing expectations, not with parameter estimation, this can be done without loss of generality. For example, if the original exponential family is curved (e.g., by parameter tying), for the purpose of computing expectations one can always work in the over-complete parameterization, and then project back to the coarse sufficient statistic for parameter estimation.

## 2.2 Markov random field reformulation

We start by constructing an equivalent but more convenient exponential family. This general construction has an associated bipartite Markov Random Field (MRF) with structure $K_{I,J}$, shown in Figure 1. This new bipartite structure should not be confused with the bipartite graph from the $K_{N,N}$ bipartite graph specific to the BM example: the former is part of the general theory, the latter is specific to the bipartite matching example.

The bipartite MRF has $I$ random variables in the first graph component, $B_1, \ldots, B_I$, each having a copy of $\mathcal{X}$ as its domain. In the second component, the graph has $J$ random variables, $S_1, \ldots, S_J$, where $S_j$ has a binary domain $\{0, 1\}$. The pairwise potential between an event $\{B_i = x\}$ in the first component and one $\{S_j = s\}$ in the second is given by $\Psi_{i,j}(x, s) = \mathbf{1}[\phi_j(x) = s]$. The following one-node potentials are also included: $\Psi_i(x) = \nu_i(x)$ and $\Psi_j(s) = e^{\theta_j s}$.

The equivalence between the two formulations follows from the rich sufficient statistic condition, which implies (for a full proof of the equivalence, see Appendix A.1):

$$\sum_{s_1 \in \{0,1\}} \sum_{s_2 \in \{0,1\}} \cdots \sum_{s_J \in \{0,1\}} \prod_{i=1}^{I} \prod_{j=1}^{J} \mathbf{1}[\phi_j(x_i) = s_j] = \begin{cases} 1 & \text{if } x_1 = x_2 = \cdots = x_I \\ 0 & \text{otherwise.} \end{cases} \tag{3}$$

This transformation into an equivalent MRF reveals several possible variational approximations. We show in the next section how loopy belief propagation [18] can be modified to tractably accommodate this transformed exponential family, even though some nodes in the graphical model—the $B_i$s—have a domain of exponential size. We then describe similar updates for mean field [19] and tree-reweighted [20] variational algorithms. We will refer to these algorithms as BPMF (Belief Propagation on Measure Factorizations), MFMF (Mean Field on Measure Factorizations) and TRWMF (Tree-Reweighted updates on Measure Factorizations). In contrast to BPMF, MFMF is guaranteed to converge[2], and TRWBF is guaranteed to provide an upper bound on the partition function.[3]

## 2.3 Implicit message representation

The variables $B_i$ have a domain of exponential size, hence if we applied belief propagation updates naively, the messages going from $B_i$ to $S_j$ would require summing over an exponential number of terms, and messages going from $S_j$ to $B_i$ would require an exponential amount of storage. To avoid summing explicitly over exponentially many terms, we adapt an idea from [7] and exploit the fact

that an efficient algorithm is assumed for computing the super-partition function $A_i$ and its derivatives. To avoid the exponential storage of messages going to $B_i$, we use an implicit representation of these messages in the canonical parameter space.

Let us denote the messages going from $S_j$ to $B_i$ by $M_{j\to i}(s), s \in \{0,1\}$ and the reverse messages by $m_{i\to j}(x), x \in \mathcal{X}$. From the definitions of $\Psi_{i,j}, \Psi_i, \Psi_j$, the explicit belief propagation updates are:

$$m_{i\to j}(s) \propto \sum_{x\in\mathcal{X}} \mathbf{1}[\phi_j(x) = s]\nu_i(x) \prod_{j':j'\neq j} M_{j'\to i}(x)$$

$$M_{j\to i}(x) \propto \sum_{s\in\{0,1\}} e^{\theta_j s}\mathbf{1}[\phi_j(x) = s] \prod_{i':i'\neq i} m_{i'\to j}(s). \tag{4}$$

The task is to get an update equation that does not represent $M_{j\to i}(x)$ explicitly, by exploiting the fact that the super-partition functions $A_i$ and their derivatives can be computed efficiently. To do so, it is convenient to use the following equivalent representation for the messages $m_{i\to j}(s)$:
$\zeta_{i,j} = \log m_{i\to j}(1) - \log m_{i\to j}(0) \in [-\infty, +\infty].$[4]

If we also let $f_{i,j}(x)$ denote any function proportional to $\prod_{j':j'\neq j} M_{j'\to i}(x)$, we can write:

$$\zeta_{i,j} = \log\left(\frac{\sum_{x\in\mathcal{X}}\phi_j(x)f_{i,j}(x)\nu_i(x)}{\sum_{x\in\mathcal{X}}(1-\phi_j(x))f_{i,j}(x)\nu_i(x)}\right) = \text{logit}\left(\frac{\sum_{x\in\mathcal{X}}\phi_j(x)f_{i,j}(x)\nu_i(x)}{\sum_{x\in\mathcal{X}}f_{i,j}(x)\nu_i(x)}\right), \tag{5}$$

where $\text{logit}(x) = \log x - \log(1-x)$. This means that if we can find a parameter vector $\boldsymbol{\xi}_{i,j} \in \mathbb{R}^J$ such that

$$f_{i,j}(x) = \exp\langle\boldsymbol{\phi}(x), \boldsymbol{\xi}_{i,j}\rangle \propto \prod_{j':j'\neq j} M_{j'\to i}(x),$$

then we could write $\zeta_{i,j} = \text{logit}\left(\nabla_j A_i(\boldsymbol{\xi}_{i,j})\right)$. We derive such a vector $\boldsymbol{\xi}_{i,j}$ as follows:

$$\prod_{j':j'\neq j} M_{j'\to i}(x) = \prod_{j':j'\neq j}\sum_{s_{j'}\in\{0,1\}} e^{\theta_{j'}s_{j'}}\mathbf{1}[\phi_{j'}(x) = s_{j'}] \prod_{i':i'\neq i} m_{i'\to j'}(s_{j'})$$

$$= \prod_{j':j'\neq j} e^{\theta_{j'}\phi_{j'}(x)} \prod_{i':i'\neq i} m_{i'\to j'}(\phi_{j'}(x))$$

$$\propto \exp\left\{\sum_{j':j'\neq j}\phi_{j'}(x)\left(\theta_{j'} + \sum_{i':i'\neq i}\zeta_{i',j'}\right)\right\},$$

where in the last step we have used the assumption that $\phi_j$ has domain $\{0,1\}$, which implies that $m_{i\to j}(\phi_j(x)) = \exp\{\phi_j(x)\log m_{i\to j}(1) + (1 - \phi_j(x))\log m_{i\to j}(0)\} \propto \exp\{\phi_j(x)\zeta_{i,j}\}$. The required parameters are therefore: $\left(\boldsymbol{\xi}_{i,j}\right)_{j'} = \mathbf{1}[j \neq j']\left(\theta_{j'} + \sum_{i':i'\neq i}\zeta_{i',j'}\right)$.

## 2.4 Reuse of partition function computations

Naively, the updates derived so far would require computing each super-partition function $J$ times at each message passing iteration. We show that this can be reduced to computing each super-partition function only once per iteration, a considerable gain.

We first define the vectors:

$$\bar{\boldsymbol{\xi}}_i = \boldsymbol{\theta} + \sum_{i':i'\neq i}\boldsymbol{\zeta}_{i'},$$

and then rewrite the numerator inside the logit function in Equation (5) as follows:

$$\sum_{x\in\mathcal{X}}\phi_j(x)f_{i,j}(x)\nu_i(x) = \sum_{s\in\{0,1\}}\sum_{x:\phi_j(x)=s}\exp\{\langle\phi(x),\bar{\boldsymbol{\xi}}_i\rangle\}\cdot e^{-\bar{\xi}_{i,j}s}\cdot s\cdot\nu_i(x)$$

$$= e^{A_i(\bar{\boldsymbol{\xi}}_i)-\bar{\xi}_{i,j}}\nabla_j A_i(\bar{\boldsymbol{\xi}}_i),$$

[4]In what follows, we will assume that $\zeta_{i,j} \in (-\infty, +\infty)$. The extended real line is treated in Appendix C.1.

and similarly for the denominator:

$$\sum_{x\in\mathcal{X}} f_{i,j}(x)\nu_i(x) = e^{A_i(\bar{\boldsymbol{\xi}}_i)-\bar{\xi}_{i,j}}\nabla_j A_i(\bar{\boldsymbol{\xi}}_i) + e^{A_i(\bar{\boldsymbol{\xi}}_i)}(1-\nabla_j A_i(\bar{\boldsymbol{\xi}}_i))$$

$$= e^{A_i(\bar{\boldsymbol{\xi}}_i)}\left(1 + (e^{-\bar{\xi}_{i,j}}-1)\nabla_j A_i(\bar{\boldsymbol{\xi}}_i)\right).$$

After plugging in the reparameterization of the numerator and denominator back into the logit function in Equation (5) and doing some algebra, we obtain the more efficient update $\zeta_{i,j} = \text{logit}\left(\nabla A_i(\bar{\xi}_{i,j})\right) - \bar{\xi}_{i,j}$, where the logit function of a vector, logit $\boldsymbol{v}$, is defined as the vector of the logit function applied to each entry of the vector $\boldsymbol{v}$. See Figure 1 for a summary of the BPMF algorithm.

## 2.5 Other variational algorithms

The ideas used to derive the BPMF updates can be extended to other variational algorithms with minor modifications. We sketch here two examples: a naive mean field algorithm, and a TRW approximation. See Appendix A.2 for details.

In the case of naive mean field applied the graphical model described in Section 2.2, the updates take a form similar to Equations (4), except that the reverse incoming message is not omitted when computing an outgoing message. As a consequence, the updates are not directional and can be associated to nodes in the graphical model rather than edges:

$$M_j(s) \propto \sum_{x\in\mathcal{X}} \mathbf{1}[\phi_j(x)=s]\nu_i(x)\prod_j m_i(x)$$

$$m_i(x) \propto \sum_{s\in\{0,1\}} e^{\theta_j s}\mathbf{1}[\phi_j(x)=s]\prod_i M_j(s).$$

This yields the following implicit updates:[5]

$$\boldsymbol{\xi}^{(t)} = \boldsymbol{\theta} + \sum_i \boldsymbol{\zeta}_i^{(t-1)}$$

$$\boldsymbol{\zeta}_i^{(t)} = \text{logit}\left(\nabla A_i\left(\boldsymbol{\xi}^{(t)}\right)\right), \tag{6}$$

and the moment approximation $\hat{\boldsymbol{\mu}} = \text{logistic}(\boldsymbol{\xi})$.

In the case of TRW, lines 3 and 6 in the pseudocode of Figure 1 stay the same, while the update in line 4 becomes:

$$\left(\boldsymbol{\xi}_{i,j}\right)_{j'} = \left(\theta_{j'} - \rho_{i\to j'}\zeta_{i,j'} + \sum_{i':i'\neq i}\rho_{i'\to j'}\zeta_{i',j'}\right)\cdot\begin{cases}\rho_{j'\to i} & \text{if } j'\neq j \\ (1-\rho_{i\to j}) & \text{otherwise,}\end{cases} \tag{7}$$

where $\rho_{i\to j}$ are marginals of a spanning tree distribution over $K_{I,J}$. We show in Appendix A.2 how the idea in Section 2.4 can be exploited to reuse computations of super-partition functions in the case of TRW as well.

## 2.6 Large factorizations

In some cases, it might not be possible to write the base measure as a succinct product of factors. Fortunately, there is a simple and elegant workaround to this problem that retains good theoretical guarantees. The basic idea is that dropping measures with domain $\{0,1\}$ in a factorization can only increase the value of the partition function. This solution is especially attractive in the context of outer approximations such as the TRW algorithm, because it preserves the upper bound property of the approximation. We show an example of this in Section 3.2.

## 3 Examples of factorizations

In this section, we show three examples of measure factorizations. See Appendix B for two more examples (partitions of the plane, and traveling salesman problems).

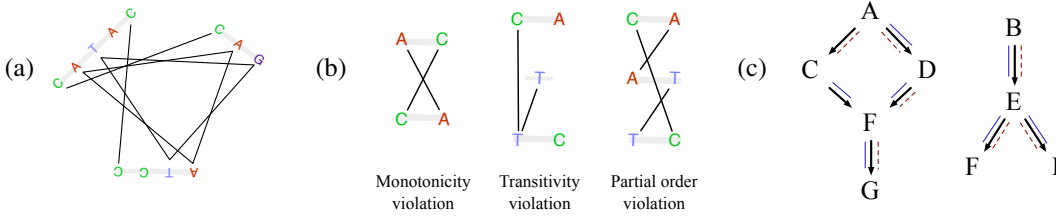

Figure 2: (a) An example of a valid multiple alignment between three sequences. (b) Examples of *invalid* multiple sequence alignments illustrating what is left out by the factors in the decomposition of Section 3.2. (c) The DAG representation of a partial order. An example of linearization is A,C,D,B,E,F,G,H,I. The fine red dashed lines and blue lines demonstrate an example of two forests covering the set of edges, forming a measure decomposition with two factors. The linearization A,D,B,E,F,G,H,I,C is an example of a state allowed by one factor but not the other.

## 3.1 More matchings

Our approach extends naturally to matchings with higher-order (augmented) sufficient statistic, and to non-bipartite/non-perfect matchings. Let us first consider an Higher-order Bipartite Model (HBM), which has all the basic sufficient statistic coordinates found in SBM, plus those of the form $\phi_j(x) = x_{m,n} \cdot x_{m+1,n+1}$. We claim that with the factorization of Equation (2), the super-partition functions $A_1$ and $A_2$ are still tractable in HBM. To see why, note that computing $A_1$ can be done by building an *auxiliary exponential family* with associated graphical model given by a chain of length $N$, and where the state space of each node in this chain is $\{1, 2, \ldots, N\}$. The basic sufficient statistic coordinates $\phi_j(x) = x_{m,n}$ are encoded as node potentials, and the augmented ones as edge potentials in the chain. This yields a running time of $O(N^3)$ for computing one super-partition function and its gradient (see Appendix A.3 for details). The auxiliary exponential family technique used here is reminiscent of [21].

Extension to non-perfect and non-bipartite matchings can also be done easily. In the first case, a dummy "null" node is added to each bipartite component. In the second case, where the original space is the set of $\binom{N}{2}$ alignment indicators, we propose a decomposition into $N$ measures. Each one checks that a single node is connected to at most one other node: $\nu_n(x) = \mathbf{1}[\sum_{n'=1}^N x_{n,n'} \leq 1]$.

## 3.2 Multiple sequence alignment

We start by describing the space of pairwise alignments (which is tractable), and then discuss the extension to multiple sequences (which quickly becomes infeasible as the number of sequences increases). Consider two sequences of length $M$ and $N$ respectively. A pairwise sequence alignment is a bipartite graph on the characters of the two sequences (where each bipartite component has the characters of one of the sequences) constrained to be *monotonic*: if a character at index $m \in \{1, \ldots, M\}$ is aligned to a character at index $n \in \{1, \ldots, N\}$ and another character at index $m' > m$ is aligned to index $n'$, then we must have $n' > n$. A multiple alignment between $K$ sequences of lengths $N_1, N_2, \ldots, N_K$ is a $K$-partite graph, where the $k$-th components' vertices are the characters of the $k$-th sequence, and such that the following three properties hold: (1) each pair of components forms a pairwise alignment as described above; (2) the alignments are *transitive*, i.e., if character $c_1$ is aligned to $c_2$ and $c_2$ is aligned to $c_3$ then $c_1$ must be aligned to $c_3$; (3) the alignments satisfy a *partial order* property: there exists a partial order $p$ on the connected components of the graph with the property that if $C_1 <_p C_2$ are two distinct connected components and $c_1 \in C_1, c_2 \in C_2$ are in the same sequence, then the index of $c_1$ in the sequence is smaller than the index of $c_2$. See Figure 2(a,b) for an illustration.

We use the technique of Section 2.6, and include only the pairwise alignment and transitivity constraints, creating a variational objective function that is an outer bound of the original objective. In this factorization, there are $\binom{K}{2}$ pairwise alignment measures, and $T =$

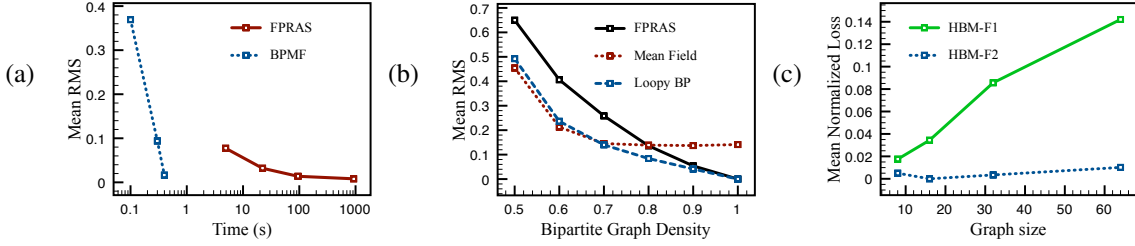

Figure 3: Experiments discussed in Section 4.1 on two of the matching models discussed. (a) and (b) on SBM, (c), on HBM.

$\sum_{k,k',k'':k\neq k'\neq k''\neq k} N_k N_{k'} N_{k''}$ transitivity measures. We show in Appendix A.4 that all the messages for one iteration can be computed in time $O(T)$.

### 3.3 Linearization of partial orders

A *linearization* of a partial order $p$ over $N$ objects is a total order $t$ over the same objects such that $x \leq_p y \Rightarrow x \leq_t y$. Counting the number of linearizations is a well-known #P problem [22]. Equivalently, the problem can be view as a matching between a DAG $G = (V, E)$ and the integers $\{1, 2, \ldots, N\}$ with the order constraints specified on the edges of the DAG.

To factorize the base measure, consider a collection of $I$ directed forests on $V$, $G_i = (V, E_i), i \in I$ such that their union covers $G$: $\cup_i E_i = E$. See Figure 2(c) for an example. For a single forest $G_i$, a straightforward generalization of the algorithm used to compute HBM's super-partition can be used. This generalization is simply to use sum-product with graphical model $G_i$ instead of sum-product on a chain as in HBM (see Appendix A.5 for details). Again, the state space of the node of the graphical model is $\{1, 2, \ldots, N\}$, but this time the edge potentials enforce the ordering constraints of the current forest.

## 4 Experiments

### 4.1 Matchings

As a first experiment, we compared the approximation of SBM described in Section 2 to the Fully Polynomial Randomized Approximation Scheme (FPRAS) described in [23]. We performed all our experiments on 100 iid random bipartite graphs of size $N$, where each edge has iid appearance probability $p$, a random graph model that we denote by $\text{RB}(N, p)$. In the first and second experiments, we used $\text{RB}(10, 0.9)$. In this case, exact computation is still possible, and we compared the mean Root Mean Squared (RMS) of the estimated moments to the truth. In Figure 3(a), we plot this quantity as a function of the time spent to compute the 100 approximations. In the variational approximation, we measured performance at each iteration of BPMF, and in the sampling approach, we measured performance after powers of two sampling rounds. The conclusion is that the variational approximation attains similar levels of error in at least one order of magnitude less time in the $\text{RB}(10, 0.9)$ regime.

Next, we show in Figure 3(b) the behavior of the algorithms as a function of $p$, where we also added the mean field algorithm to the comparison. In each data point in the graph, the FPRAS was run no less than one order of magnitude more time than the variational algorithms. Both variational strategies outperform the FPRAS in low-density regimes, where mean field also slightly outperforms BPMF. On the other hand, for high-density regimes, only BPMF outperforms the FPRAS, and mean field has a bias compared to the other two methods.

The third experiment concerns the augmented matching model, HBM. Here we compare two types of factorization and investigate the scalability of the approaches to larger graphs. Factorization F1 is a simpler factorization of the form described in Section 3.1 for non-bipartite graphs. This ignores the higher-order sufficient statistic coordinates, creating an outer approximation. Factorization F2,

| | Sum of Pairs score (SP) | | | | |
|---|---|---|---|---|---|
| BAliBASE protein group | BPMF-1 | BPMF-2 | BPMF-3 | Clustal [24] | ProbCons [25] |
| short, $< 25\%$ identity | 0.68 | 0.74 | **0.76** | 0.71 | 0.72 |
| short, $20\% - 40\%$ identity | 0.94 | **0.95** | **0.95** | 0.89 | 0.92 |
| short, $> 35\%$ identity | 0.97 | **0.98** | **0.98** | 0.97 | **0.98** |
| All | 0.88 | **0.91** | **0.91** | 0.88 | 0.89 |

Table 1: Average SP scores in the ref1/test1 directory of BAliBASE. BPMF-$i$ denotes the average SP of the BPMF algorithm after $i$ iterations of (parallel) message passing.

described in Section 3.1 specifically for HBM, is tighter. The experimental setup is based on a generative model over noisy observations of bipartite perfect matchings described in Appendix C.2. We show in Figure 3(c) the results of a sequence of these experiments for different bipartite component sizes $N/2$. This experiments demonstrates the scalability of sophisticated factorizations, and their superiority over simpler ones.

## 4.2 Multiple sequence alignment

To assess the practical significance of this framework, we also apply it to BAliBASE [6], a standard protein multiple sequence alignment benchmark. We compared our system to Clustal 2.0.12 [24], the most popular multiple alignment tool, and ProbCons 1.12, a state-of-the-art system [25] that also relies on enforcing transitivity constraints, but which is not derived via the optimization of an objective function. Our system uses a basic pair HMM [26] to score pairwise alignments. This scoring function captures a proper subset of the biological knowledge exploited by Clustal and ProbCons.[6] The advantage of our system over the other systems is the better optimization technique, based on the measure factorization described in Section 3.2. We used a standard technique to transform the pairwise alignment marginals into a single valid multiple sequence alignment (see Appendix C.3). Our system outperformed both baselines after three BPMF parallel message passing iterations. The algorithm converged in all protein groups, and performance was identical after more than three iterations. Although the overall performance gain is not statistically significant according to a Wilcoxon signed-rank test, the larger gains were obtained in the small identity subset, the "twilight zone" where research on multiple sequence alignment has focused.

One caveat of this multiple alignment approach is its running time, which is cubic in the length of the longest sequence, while most multiple sequence alignment approaches are quadratic. For example, the running time for one iteration of BPMF in this experiment was 364.67s, but only 0.98s for Clustal—this is why we have restricted the experiments to the short sequences section of BAliBASE. Fortunately, several techniques are available to decrease the computational complexity of this algorithm: the transitivity factors can be subsampled using a coarse pass, or along a phylogenetic tree; and computation of the factors can be entirely parallelized. These improvements are orthogonal to the main point of this paper, so we leave them for future work.

## 5 Conclusion

Computing the moments of discrete exponential families can be difficult for two reasons: the structure of the sufficient statistic that can create junction trees of high tree-width, and the structure of the base measures that can induce an intractable combinatorial space. Most previous work on variational approximations has focused on the first difficulty; however, the second challenge also arises frequently in machine learning. In this work, we have presented a framework that fills this gap. It is based on an intuitive notion of measure factorization, which, as we have shown, applies to a variety of combinatorial spaces. This notion enables variational algorithms to be adapted to the combinatorial setting. Our experiments both on synthetic and naturally-occurring data demonstrate the viability of the method compared to competing state-of-the-art algorithms.

## Footnotes

[1]The appendices can be found in the supplementary material.

[2]Although we did not have convergence issues with BPMF in our experiments.

[3]Surprisingly, MFMF does not provide a lower bound (see Appendix A.6).

[5]Assuming that naive mean field is optimized coordinate-wise, with an ordering that optimizes all of the $m_i$'s, then all of the $M_j$'s.

[6]More precisely it captures long gap and hydrophobic core modeling.

# References

[1] Alexander Karzanov and Leonid Khachiyan. On the conductance of order Markov chains. *Order*, V8(1):7–15, March 1991.

[2] Mark Jerrum, Alistair Sinclair, and Eric Vigoda. A polynomial-time approximation algorithm for the permanent of a matrix with non-negative entries. In *Proceedings of the Annual ACM Symposium on Theory of Computing*, pages 712–721, 2001.

[3] David Wilson. Mixing times of lozenge tiling and card shuffling Markov chains. *The Annals of Applied Probability*, 14:274–325, 2004.

[4] Adam Siepel and David Haussler. Phylogenetic estimation of context-dependent substitution rates by maximum likelihood. *Mol Biol Evol*, 21(3):468–488, 2004.

[5] Martin J. Wainwright and Michael I. Jordan. Graphical models, exponential families, and variational inference. *Foundations and Trends in Machine Learning*, 1:1–305, 2008.

[6] Julie Thompson, Frédéric Plewniak, and Olivier Poch. BAliBASE: A benchmark alignments database for the evaluation of multiple sequence alignment programs. *Bioinformatics*, 15:87–88, 1999.

[7] David A. Smith and Jason Eisner. Dependency parsing by belief propagation. In *Proceedings of the Conference on Empirical Methods in Natural Language Processing (EMNLP)*, pages 145–156, Honolulu, October 2008.

[8] David Burkett, John Blitzer, and Dan Klein. Joint parsing and alignment with weakly synchronized grammars. In *North American Association for Computational Linguistics*, Los Angeles, 2010.

[9] Bert Huang and Tony Jebara. Approximating the permanent with belief propagation. *ArXiv e-prints*, 2009.

[10] Yusuke Watanabe and Michael Chertkov. Belief propagation and loop calculus for the permanent of a non-negative matrix. *J. Phys. A: Math. Theor.*, 2010.

[11] Ben Taskar, Dan Klein, Michael Collins, Daphne Koller, and Christopher Manning. Max-margin parsing. In *EMNLP*, 2004.

[12] Ben Taskar, Simon Lacoste-Julien, and Dan Klein. A discriminative matching approach to word alignment. In *EMNLP 2005*, 2005.

[13] John Duchi, Daniel Tarlow, Gal Elidan, and Daphne Koller. Using combinatorial optimization within max-product belief propagation. In *Advances in Neural Information Processing Systems*, 2007.

[14] Aron Culotta, Andrew McCallum, Bart Selman, and Ashish Sabharwal. Sparse message passing algorithms for weighted maximum satisfiability. In *New England Student Symposium on Artificial Intelligence*, 2007.

[15] Percy Liang, Ben Taskar, and Dan Klein. Alignment by agreement. In *North American Association for Computational Linguistics (NAACL)*, pages 104–111, 2006.

[16] Percy Liang, Dan Klein, and Michael I. Jordan. Agreement-based learning. In *Advances in Neural Information Processing Systems (NIPS)*, 2008.

[17] Leslie G. Valiant. The complexity of computing the permanent. *Theoret. Comput. Sci.*, 1979.

[18] Jonathan S. Yedidia, William T. Freeman, and Yair Weiss. Generalized belief propagation. In *Advances in Neural Information Processing Systems*, pages 689–695, Cambridge, MA, 2001. MIT Press.

[19] Carsten Peterson and James R. Anderson. A mean field theory learning algorithm for neural networks. *Complex Systems*, 1:995–1019, 1987.

[20] Martin J. Wainwright, Tommi S. Jaakkola, and Alan S. Willsky. Tree-reweighted belief propagation algorithms and approximate ML estimation by pseudomoment matching. In *Proceedings of the International Conference on Articial Intelligence and Statistics*, 2003.

[21] Alexandre Bouchard-Côté and Michael I. Jordan. Optimization of structured mean field objectives. In *Proceedings of Uncertainty in Artifical Intelligence*, 2009.

[22] Graham Brightwell and Peter Winkler. Counting linear extensions. *Order*, 1991.

[23] Lars Eilstrup Rasmussen. Approximating the permanent: A simple approach. *Random Structures and Algorithms*, 1992.

[24] Des G. Higgins and Paul M. Sharp. CLUSTAL: a package for performing multiple sequence alignment on a microcomputer. *Gene*, 73:237–244, 1988.

[25] Chuong B. Do, Mahathi S. P. Mahabhashyam, Michael Brudno, and Serafim Batzoglou. PROBCONS: Probabilistic consistency-based multiple sequence alignment. *Genome Research*, 15:330–340, 2005.

[26] David B. Searls and Kevin P. Murphy. Automata-theoretic models of mutation and alignment. In *Proc Int Conf Intell Syst Mol Biol.*, 1995.

